# Exact inference and learning for cumulative distribution functions on loopy graphs

**Jim C. Huang, Nebojsa Jojic and Christopher Meek**
Microsoft Research
One Microsoft Way, Redmond, WA 98052

## Abstract

Many problem domains including climatology and epidemiology require models that can capture both heavy-tailed statistics and local dependencies. Specifying such distributions using graphical models for probability density functions (PDFs) generally lead to intractable inference and learning. Cumulative distribution networks (CDNs) provide a means to tractably specify multivariate heavy-tailed models as a product of cumulative distribution functions (CDFs). Existing algorithms for inference and learning in CDNs are limited to those with tree-structured (non-loopy) graphs. In this paper, we develop inference and learning algorithms for CDNs with arbitrary topology. Our approach to inference and learning relies on recursively decomposing the computation of mixed derivatives based on a junction trees over the cumulative distribution functions. We demonstrate that our systematic approach to utilizing the sparsity represented by the junction tree yields significant performance improvements over the general symbolic differentiation programs Mathematica and D*. Using two real-world datasets, we demonstrate that non-tree structured (loopy) CDNs are able to provide significantly better fits to the data as compared to tree-structured and unstructured CDNs and other heavy-tailed multivariate distributions such as the multivariate copula and logistic models.

## 1 Introduction

The last two decades have been marked by significant advances in modeling multivariate probability density functions (PDFs) on graphs. Various inference and learning algorithms have been successfully developed that take advantage of known variable dependence which can be used to simplify computations and avoid overtraining. A major source of difficulty for such algorithms is the need to compute a normalization term, as graphical models generally assume a factorized form for the joint PDF. To make these models tractable, the factors themselves can be chosen to have tractable forms such as Gaussians. Such choices may then make the model unsuitable for many types of data, such as data with heavy-tailed statistics that are a quintessential feature in many application areas such as climatology and epidemiology. Recently, a number of techniques have been proposed to allow for both heavy-tailed/non-Gaussian distributions with a specifiable variable dependence structure. Most of these methods are based on transforming the data to make it more easily modeled by Gaussian PDF-fitting techniques, an example of which is the Gaussian copula [11] parameterized as a CDF defined on nonlinearly transformed variables. In addition to copula models, many non-Gaussian distributions are conveniently parameterized as CDFs [2]. Most existing CDF models, however, do not allow the specification of local dependence structures and thus can only be applied to very low-dimensional problems.

Recently, a class of multiplicative CDF models has been proposed as a way of modeling structured CDFs. The cumulative distribution networks (CDNs) model a multivariate CDF as a product over functions, each dependent on a small subset of variables and each having a CDF form [6, 7]. One of the key advantages of this approach is that it eliminates the need to enforce normalization constraints that complicate inference and learning in graphical models of PDFs. An example of a CDN is shown in Figure 1(a), where diamonds correspond to CDN functions and circles represent variables. In a CDN, inference and learning involves computation of derivatives of the joint CDF with respect to model variables and parameters. The graphical model then allows us to efficiently perform inference and learning for non-loopy CDNs using message-passing [6, 8]. Models of this form have

been applied to multivariate heavy-tailed data in climatology and epidemiology where they have demonstrated improved predictive performance as compared to several graphical models for PDFs despite the restriction to tree-structured CDNs. Non-loopy CDNs may however be limited models and adding functions to the CDN may provide significantly more expressive models, with the caveat that the resulting CDN may become loopy and previous algorithms for inference and learning in CDNs then cease to be exact.

Our aim in this paper is to provide an effective algorithm for learning and inference in loopy CDNs, thus improving on previous approaches which were limited to CDNs with non-loopy dependencies. In principle, symbolic differentiation algorithms such as Mathematica [16] and D* [4] could be used for inference and learning for loopy CDNs. However, as we demonstrate, such generic algorithms quickly become intractable for larger models. In this paper, we develop the JDiff algorithm which uses the graphical structure to simplify the computation of the derivative and enables both inference and learning for CDNs of arbitrary topology. In addition, we provide an analysis of the time and space complexity of the algorithm and provide experiments comparing JDiff to Mathematica and D*, in which we show that JDiff runs in less time and can handle significantly larger graphs. We also provide an empirical comparison of several methods for modeling multivariate distributions as applied to rainfall data and H1N1 data. We show that loopy CDNs provide significantly better model fits for multivariate heavy-tailed data than non-loopy CDNs. Furthermore, these models outperform models based on Gaussian copulas [11], as well as multivariate heavy tailed models that do not allow for structure specification.

## 2   Cumulative distribution networks

In this section we establish preliminaries about learning and inference for CDNs [6, 7, 8]. Let $\mathbf{x}$ be a vector of observed values for random variables in the set $V$ and let $x_\alpha, \mathbf{x}_A$ denote the observed values for variable node $\alpha \in V$ and variable set $A \subseteq V$. Let $\mathcal{N}(s)$ be the set of neighboring variable nodes for function node $s$. Define the operator $\partial_{\mathbf{x}_A}[\cdot]$ as the mixed derivative operator with respect to variables in set $A$. For example, $\partial_{x_{1,2,3}}[F(x_1, x_2, x_3)] \equiv \frac{\partial^3 F}{\partial x_1 \partial x_2 \partial x_3}$. Throughout the paper we will be dealing primarily with continuous random variables and so we will generally deal with PDFs, with probability mass functions (PMFs) as a special case. We also assume in the sequel that all derivatives of a CDF with respect to any and all arguments exist and are continuous and as a result any mixed derivative of the CDF is invariant to the order of differentiation (Schwarz' theorem).

**Definition 2.1.** *The cumulative distribution network (CDN) consists of (1) an undirected bipartite graphical model consisting of a bipartite graph $\mathcal{G} = (V, S, E)$, where $V$ denotes variable nodes and $S$ denotes function nodes, with edges in $E$ connecting function nodes to variable nodes and (2) a specification of functions $\phi_s(\mathbf{x}_s)$ for each function node $s \in S$, where $\mathbf{x}_s \equiv \mathbf{x}_{\mathcal{N}(s)}$, $\cup_{s \in S} \mathcal{N}(s) = V$ and each function $\phi_s : \mathbb{R}^{|\mathcal{N}(s)|} \mapsto [0, 1]$ satisfies the properties of a CDF. The joint CDF over the variables in the CDN is then given by the product of CDFs $\phi_s$, or $F(\mathbf{x}) = \prod_{s \in S} \phi_s(\mathbf{x}_s)$, where each CDF $\phi_s$ is defined over neighboring variable nodes $\mathcal{N}(s)$.* $\square$

For example, in the CDN of Figure 1(a), each diamond corresponds to a function $\phi_s$ defined over neighboring pairs of variable nodes, such that the product of functions satisfies the properties of a CDF. In the sequel we will assume that both $F$ and CDN functions $\phi_s$ are parametric functions of parameter vector $\boldsymbol{\theta}$ and so $F \equiv F(\mathbf{x}) \equiv F(\mathbf{x}|\boldsymbol{\theta})$ and $\phi_s \equiv \phi_s(\mathbf{x}_s) \equiv \phi_s(\mathbf{x}_s; \boldsymbol{\theta})$. In a CDN, the marginal CDF for any subset $A \subseteq V$ is obtained simply by taking limits such that $F(\mathbf{x}_A) = \lim_{\mathbf{x}_{V \setminus A} \to \infty} F(\mathbf{x})$, which can be done in constant time for each variable.

### 2.1   Inference and learning in CDNs as differentiation

For a joint CDF, the problems of inference and likelihood evaluation, or computing conditional CDFs and marginal PDFs, both correspond to mixed differentiation of the joint CDF [6]. In particular, the conditional CDF $F(\mathbf{x}_B|\mathbf{x}_A)$ is related to the mixed derivative $\partial_{\mathbf{x}_A}[F(\mathbf{x}_A, \mathbf{x}_B)]$ by $F(\mathbf{x}_B|\mathbf{x}_A) = \frac{\partial_{\mathbf{x}_A}[F(\mathbf{x}_A, \mathbf{x}_B)]}{\partial_{\mathbf{x}_A}[F(\mathbf{x}_A)]}$. In the case of evaluating the likelihood corresponding to the model, we note that for CDF $F(\mathbf{x}|\boldsymbol{\theta})$, the PDF is defined as $P(\mathbf{x}|\boldsymbol{\theta}) = \partial_{\mathbf{x}}[F(\mathbf{x}|\boldsymbol{\theta})]$. In order to perform maximum-likelihood estimation, we require the gradient vector $\nabla_{\boldsymbol{\theta}} \log P(\mathbf{x}|\boldsymbol{\theta}) = \frac{1}{P(\mathbf{x}|\boldsymbol{\theta})} \nabla_{\boldsymbol{\theta}} P(\mathbf{x}|\boldsymbol{\theta})$, which requires us to compute a vector of single derivatives $\partial_{\theta_i}[P(\mathbf{x}|\boldsymbol{\theta})]$ of the joint CDF with respect to parameters in the model.

## 2.2 Message-passing algorithms for differentiation in non-loopy graphs

As described above, inference and learning in a CDN corresponds to computing derivatives of the CDF with respect to subsets of variables and/or model parameters. For inference in non-loopy CDNs, computing mixed derivatives of the form $\partial_{\mathbf{x}_A}[F(\mathbf{x})]$ for some subset of nodes $A \subseteq V$ can be solved efficiently by the derivative-sum-product (DSP) algorithm of [6]. In analogy to the way in which marginalization in graphical models for PDFs can be decomposed into a series of local computations, the DSP algorithm decomposes the global computation of the *total mixed derivative* $\partial_{\mathbf{x}}[F(\mathbf{x})]$ into a series of local computations by the passing of messages that correspond to mixed derivatives of $F(\mathbf{x})$ with respect to subsets of variables in the model. To evaluate the model likelihood, messages are passed from leaf nodes to the root variable node and the product of incoming root messages is differentiated. This procedure provably produces the correct likelihood $P(\mathbf{x}|\boldsymbol{\theta}) = \partial_{\mathbf{x}}[F(\mathbf{x}|\boldsymbol{\theta})]$ for non-loopy CDNs [6].

To estimate model parameters $\boldsymbol{\theta}$ for which the likelihood over i.i.d. data samples $\mathbf{x}_1, \cdots, \mathbf{x}_N$ is optimized, we can further make use of the gradient of the log-likelihood $\nabla_{\boldsymbol{\theta}} \log P(\mathbf{x}|\boldsymbol{\theta})$ within a gradient-based optimization algorithm. As in the DSP inference algorithm, the computation of the gradient can also be broken down into a series of local gradient computations. The gradient-derivative-product (GDP) algorithm [8] updates the *gradients* of the messages from the DSP algorithm and passes these from leaf nodes to the root variable node in the CDN, provably obtaining the correct gradient of the log-likelihood of a particular set of observations $\mathbf{x}$ for a non-loopy CDN.

## 3 Differentiation in loopy graphs

For loopy graphs, the DSP and GDP algorithms are not guaranteed to yield the correct derivative computations. For the general case of differentiating a product of CDFs, computing the total mixed derivative requires time and space exponential in the number of variables. To see this, consider the simple example of the derivative of a product of two functions $f, g$, both of which are functions of $\mathbf{x} = [x_1, \cdots, x_K]$. The mixed derivative of the product is then given by [5]

$$\partial_{\mathbf{x}}[f(\mathbf{x})g(\mathbf{x})] = \sum_{U \subseteq \{1, \cdots, K\}} \partial_{\mathbf{x}_U}[f(\mathbf{x})]\partial_{\mathbf{x}_{\{1, \cdots, K\} \setminus U}}[g(\mathbf{x})], \tag{1}$$

a summation that contains $2^K$ terms. As computing the mixed derivative of a product of more functions will entail even greater complexity, the naïve approach will in general be intractable. However, as we show in this paper, a CDN's sparse graphical structure may often point to ways to computing these derivatives efficiently, with non-loopy graphs being special, previously-studied cases. To motivate our approach, consider the following lemma that follows in straightforward fashion from the product rule of differentiation:

**Lemma 3.1.** *Let $\mathcal{G} = (V, S, E)$ be a CDN and let $F(\mathbf{x}) = \prod_{s \in S} \phi_s(\mathbf{x}_s)$ be defined over variables in $V$. Let $M_1, M_2$ be a partition of the function nodes $S$ and let $G_1(\mathbf{x}_{C_1}) = \prod_{s \in M_1} \phi_s(\mathbf{x}_s)$ and $G_2(\mathbf{x}_{C_2}) = \prod_{s \in M_2} \phi_s(\mathbf{x}_s)$, where $C_1 = \bigcup_{s \in M_1} \mathcal{N}(s)$ and $C_2 = \bigcup_{s \in M_2} \mathcal{N}(s)$ are the variables that are arguments to $G_1, G_2$. Let $S_{1,2} = C_1 \bigcap C_2$. Then*

$$\partial_{\mathbf{x}}[G_1(\mathbf{x}_{C_1})G_2(\mathbf{x}_{C_2})] = \sum_{A \subseteq S_{1,2}} \partial_{\mathbf{x}_{C_1 \setminus S_{1,2}}} \Big[\partial_{\mathbf{x}_A}[G_1(\mathbf{x}_{C_1})]\Big]\partial_{\mathbf{x}_{C_2 \setminus S_{1,2}}}\Big[\partial_{\mathbf{x}_{S_{1,2} \setminus A}}[G_2(\mathbf{x}_{C_2})]\Big]. \tag{2}$$

*Proof.* Define $L = C_1 \setminus S_{1,2}$ and $R = C_2 \setminus S_{1,2}$. Then

$$\partial_{\mathbf{x}}[F(\mathbf{x})] = \partial_{\mathbf{x}}[G_1(\mathbf{x}_{C_1})G_2(\mathbf{x}_{C_2})] = \sum_{U \subseteq V} \partial_{\mathbf{x}_U}[G_1(\mathbf{x}_{C_1})]\partial_{\mathbf{x}_{V \setminus U}}[G_2(\mathbf{x}_{C_2})]$$

$$= \sum_{A \subseteq S_{1,2}} \sum_{B \subseteq L} \sum_{C \subseteq R} \partial_{\mathbf{x}_{A,B,C}}[G_1(\mathbf{x}_{C_1})]\partial_{\mathbf{x}_{S_{1,2} \setminus A, L \setminus B, R \setminus C}}[G_2(\mathbf{x}_{C_2})]$$

$$= \sum_{A \subseteq S_{1,2}} \partial_{\mathbf{x}_{A,L}}[G_1(\mathbf{x}_{C_1})]\partial_{\mathbf{x}_{S_{1,2} \setminus A, R}}[G_2(\mathbf{x}_{C_2})]. \tag{3}$$

The last step follows from identifying all derivatives that are zero, as we note that in the above, $\partial_{\mathbf{x}_C}[G_1(\mathbf{x}_{C_1})] = 0$ for $C \neq \emptyset$ and similarly, $\partial_{\mathbf{x}_{L \setminus B}}[G_2(\mathbf{x}_{C_2})] = 0$ for $L \setminus B \neq \emptyset$. $\square$

The number of individual steps needed to complete the differentiation in (2) depends on the size of the variable intersection set $S_{1,2} = C_1 \bigcap C_2$. When the two factors $G_1, G_2$ depend on two variable

sets that do not intersect, then the differentiation can be simplified by independently computing derivatives for each factor and multiplying. For example, for the CDN in Figure 1(a), partitioning the problem such that $C_1 = \{2,3,4,6\}, C_2 = \{1,2,5,7\}$ yields a more efficient computation than the brute force approach. Significant computational advantages exist even when $S \neq \emptyset$, provided $|S_{1,2}|$ is small. This suggests that we can recursively decompose the total mixed derivative and gradient computations into a series of simpler computations so that $\partial_{\mathbf{x}}[F(\mathbf{x})]$ reduces to a sum that contains far fewer terms than that required by brute force. In such a recursion, the total product of factors is always broken into parts that share as few variables as possible. This is efficient for most CDNs of interest that consist of a large number of factors that each depend on a small subset of variables. Such a recursive decomposition is naturally represented using a junction tree [12] for the CDN in which we will pass messages corresponding to local derivative computations.

## 3.1 Differentiation in junction trees

In a CDN $\mathcal{G} = (V, S, E)$, let $\{C_1, \cdots, C_n\}$ be a set of $n$ subsets of variable nodes in $V$, where $\bigcup_{i=1}^{n} C_i = V$. Let $\mathcal{C} = \{1, \cdots, n\}$ and $\mathcal{T} = (\mathcal{E}, \mathcal{C})$ be a tree where $\mathcal{E}$ is the set of undirected edges so that for any pair $i, j \in \mathcal{C}$ there is a unique path from $i$ to $j$. Then $\mathcal{T}$ is a junction tree for $\mathcal{G}$ if any intersection $C_i \bigcap C_j$ is contained in the subset $C_k$ corresponding to a node $k$ on the path from $i$ to $j$. For each directed edge $(i, j)$ we define the separator set as $S_{i,j} = C_i \bigcap C_j$. An example of a CDN and a corresponding junction tree are shown in Figures 1(a), 1(b).

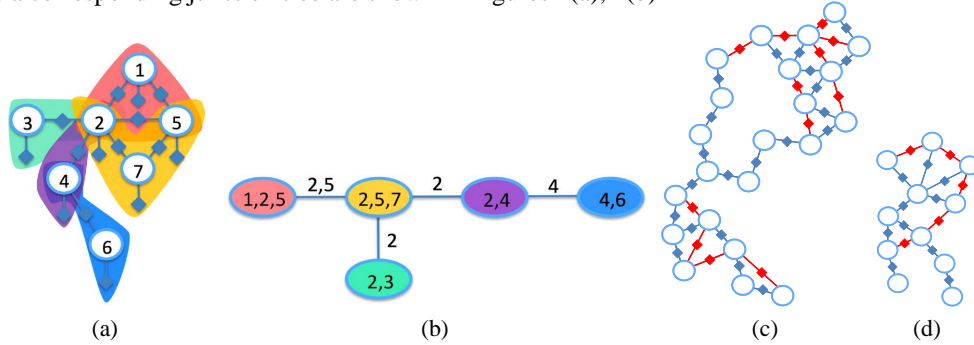

|  (a) | (b) | (c) | (d) |
| --- | --- | --- | --- |

Figure 1: a) An example of a CDN with 7 variable nodes (circles) and 15 function nodes (diamonds); b) A junction tree obtained from the CDN of a). Separating sets are shown for each edge connecting nodes in the junction tree, each corresponding to a connected subset of variables in the CDN; c), d) CDNs used to model the rainfall and H1N1 datasets. Nodes and edges in the non-loopy CDNs of [8] are shown in blue and function nodes/edges that were added to the trees are shown in red.

Since $\mathcal{T}$ is a tree, we can root the tree at some node in $\mathcal{C}$, say $r$. Given $r$, denote by $\tau_i^j$ the subset of elements of $\mathcal{C}$ that are in the subtree of $\mathcal{T}$ rooted at $j$ and containing $i$. Also, let $\mathcal{E}_i$ be the set of neighbors of $i$ in $\mathcal{T}$, such that $\mathcal{E}_i = \{j | (i, j) \in \mathcal{E}\}$. Finally, let $C_A = \bigcup_{i \in A} C_i$. Suppose $M_1, \cdots, M_n$ is a partition of $S$ such that for any $i = 1, \cdots, n$, $M_i$ consists of all $s \in S$ whose neighbors in $\mathcal{G}$ are contained in $C_i$ and there is no $j > i$ such that all neighbors of $s \in M_i$ are included in $C_j$. Define the potential function $\psi_i(\mathbf{x}_{C_i}) = \prod_{s \in M_i} \phi_s(\mathbf{x}_s)$ for subset $C_i$. We can then write the joint CDF as

$$F(\mathbf{x}) = \psi_r(\mathbf{x}_{C_r}) \prod_{k \in \mathcal{E}_r} T_k^r(\mathbf{x}), \tag{4}$$

where $T_k^r(\mathbf{x}) = \prod_{j \in \tau_k^r} \psi_j(\mathbf{x}_{C_j})$, with $\psi_j$ defined as above. Computing the probability $P(\mathbf{x})$ then corresponds to computing

$$\partial_{\mathbf{x}}\left[\psi_r(\mathbf{x}_{C_r}) \prod_{k \in \mathcal{E}_r} T_k^r(\mathbf{x})\right] = \partial_{\mathbf{x}_{C_r}}\left[\psi_r(\mathbf{x}_{C_r}) \prod_{k \in \mathcal{E}_r} \partial_{\mathbf{x}_{C_{\tau_k^r} \setminus S_{r,k}}}[T_k^r(\mathbf{x})]\right]$$

$$= \partial_{\mathbf{x}_{C_r}}\left[\psi_r(\mathbf{x}_{C_r}) \prod_{k \in \mathcal{E}_r} m_{k \to r}(\emptyset)\right], \tag{5}$$

where we have defined messages $m_{k \to r}(A) \equiv \partial_{\mathbf{x}_A}\left[\partial_{\mathbf{x}_{C_{\tau_k^r} \setminus S_{r,k}}}[T_k^r(\mathbf{x})]\right]$, with $m_{k \to r}(\emptyset) = \partial_{\mathbf{x}_{C_{\tau_k^r} \setminus S_{r,k}}}[T_k^r(\mathbf{x})]$. It remains to determine how we can efficiently compute messages in the above expression. We notice that for *any* given $i \in \mathcal{C}$ with $A \subseteq C_i$ and $U_i \subseteq \mathcal{E}_i$, we can define the

quantity $m_i(A, U_i) \equiv \partial_{\mathbf{x}_A}\left[\psi_i(\mathbf{x}_{C_i})\prod_{j\in U_i}m_{j\to i}(\emptyset)\right]$. Now select $k \in U_i$ for the given $i$: we can recursively re-write the above as

$$m_i(A, U_i) = \partial_{\mathbf{x}_A}\left[\left(\psi_i(\mathbf{x}_{C_i})\prod_{j\in U_i\setminus k}m_{j\to i}(\emptyset)\right)m_{k\to i}(\emptyset)\right] = \partial_{\mathbf{x}_A}\left[m_i(\emptyset, U_i\setminus k)m_{k\to i}(\emptyset)\right]$$

$$= \sum_{B\subseteq A}m_{k\to i}(B)m_i(A\setminus B, U_i\setminus k) = \sum_{B\subseteq A\bigcap S_{i,k}}m_{k\to i}(B)m_i(A\setminus B, U_i\setminus k), \quad (6)$$

where in the last step we note that whenever $B\bigcap S_{i,k}=\emptyset$, $m_{k\to i}(B)=0$, since by definition message $m_{k\to i}(A)$ does not depend on variables in $C_i\setminus S_{i,k}$. From the definition of message $m_{j\to i}(A)$, for any $A\subseteq S_{i,j}$ we also have

$$m_{j\to i}(A) = \partial_{\mathbf{x}_A}\left[\partial_{\mathbf{x}_{C_{\tau_j^i}\setminus S_{i,j}}}[T_j^i(\mathbf{x})]\right] = \partial_{\mathbf{x}_{A,C_j\setminus S_{i,j}}}\left[\psi_j(\mathbf{x}_{C_j})\prod_{l\in\mathcal{E}_j\setminus i}\partial_{\mathbf{x}_{C_{\tau_l^j}\setminus S_{l,j}}}[T_l^j(\mathbf{x})]\right]$$

$$= m_j\left(A\bigcup C_j\setminus S_{i,j}, \mathcal{E}_j\setminus i\right), \quad (7)$$

where $\tau_l^j$ is the subtree of $\mathcal{T}$ rooted at $j$ and containing $l$. Thus, we can recursively compute functions $m_i, m_{j\to i}$ by applying the above updates for each node in $\mathcal{T}$, starting from from leaf nodes of $\mathcal{T}$ and up to the root node $r$. At the root node, the correct mixed derivative is then given by $P(\mathbf{x}) = \partial_{\mathbf{x}}[F(\mathbf{x})] = m_r(C_r, \mathcal{E}_r)$. Note that the messages can be kept in a symbolic form as functions over appropriate variables, or, as is the case in the experiments section, they can simply be evaluated for the given data $\mathbf{x}$. In the latter case, each message reduces to a scalar, as we can evaluate derivatives of the functions in the model for fixed $\mathbf{x}, \boldsymbol{\theta}$ and so we do not need to store increasingly complex symbolic terms.

### 3.2 Maximum-likelihood learning in junction trees

While computing $P(\mathbf{x}|\boldsymbol{\theta}) = \partial_{\mathbf{x}}[F(\mathbf{x}|\boldsymbol{\theta})]$, we can in parallel obtain the gradient of the likelihood function. The likelihood is equal to the message $m_r(C_r, \mathcal{E}_r)$ at the root node $r\in\mathcal{T}$. The computation of its gradient $\nabla_{\boldsymbol{\theta}}m_r(C_r, \mathcal{E}_r)$ can be decomposed in a similar fashion to the decomposition of the mixed derivative computation. The gradient of each message $m_i, m_{j\to i}$ in the junction tree decomposition is updated in parallel with the likelihood messages through the use of gradient messages $\mathbf{g}_i\equiv\nabla_{\boldsymbol{\theta}}m_i$ and $\mathbf{g}_{j\to i}\equiv\nabla_{\boldsymbol{\theta}}m_{j\to i}$.

The algorithm for computing both the likelihood and its gradient, which we call JDiff for *junction tree differentiation*, is shown in Algorithm 1. Thus by recursively computing the messages and their gradients starting from leaf nodes of $\mathcal{T}$ to the root node $r$, we can obtain the exact likelihood and gradient vector for the CDF modelled by $\mathcal{G}$.

### 3.3 Running time analysis

The space and time complexity of JDiff is dominated by Steps 1-3 in Algorithm 1: we quantify this in the next Theorem.

**Theorem 3.2.** *The time and space complexity of the JDiff algorithm is*

$$O\left(\max_j(|M_j|+1)^{|C_j|} + \max_{(j,k)\in\mathcal{E}}(|\mathcal{E}_j|-1)*2^{|C_j\setminus S_{j,k}|}3^{|S_{j,k}|}\right). \quad (8)$$

*Proof.* The complexity of Step 1 in Algorithm 1 is given by $\sum_{k=1}^{C_j}\binom{|C_j|}{k}|M_j|^k = O\left((M_j+1)^{|C_j|}\right)$, which is the total number of terms in the expanded sum of products form for computing mixed derivatives $\partial_{\mathbf{x}_A}[\psi_j]$ for all $A\subseteq C_j$. Step 2 has complexity bounded by

$$O\left((|\mathcal{E}_j|-1)*\max_{k\in\mathcal{E}_j}\sum_{l=0}^{S_{j,k}}\binom{|S_{j,k}|}{l}2^{|C_j\setminus S_{j,k}|}2^l\right) = (|\mathcal{E}_j|-1)*O(\max_{k\in\mathcal{E}_j}2^{|C_j\setminus S_{j,k}|}3^{|S_{j,k}|}) \quad (9)$$

since the cost of computing derivatives for each $A\subseteq C_j$ is a function of the size of the intersection with $S_{i,j}$. Thus we have the number of ways that an intersection can be of size $l$ times the number of ways that we can choose the variables not in the separator $S_{j,k}$ times the cost for that size of overlap. Finally, Step 3 has complexity bounded by $O(2^{|S_{j,k}|})$. The total time and space complexity is then of order given by $O\left(\max_j(|M_j|+1)^{|C_j|} + \max_{(j,k)\in\mathcal{E}}(|\mathcal{E}_j|-1)*2^{|C_j\setminus S_{j,k}|}3^{|S_{j,k}|}\right)$. $\square$

**Algorithm 1:** JDiff: A junction tree algorithm for computing the likelihood $\partial_{\mathbf{x}}[F(\mathbf{x}|\boldsymbol{\theta})]$ and its gradient $\nabla_{\boldsymbol{\theta}}\partial_{\mathbf{x}}[F(\mathbf{x}|\boldsymbol{\theta})]$ for a CDN $\mathcal{G}$. Lines marked 1,2,3 dominate the space and time complexity.

---

**Input**: A CDN $\mathcal{G} = (V, S, E)$, a junction tree $\mathcal{T} \equiv \mathcal{T}(\mathcal{G}) = (\mathcal{E}, \mathcal{C})$ with node set $\mathcal{C} = \{1, \cdots, n\}$ and edge set $\mathcal{E}$, where each $i \in \mathcal{C}$ indexes a subset $C_i \subseteq V$. Let $r \in \mathcal{C}$ be the root of $\mathcal{T}$ and denote the subtree of $\mathcal{T}$ rooted at $j$ containing $k$ by $\tau_k^j$. Let $M_1, \cdots, M_n$ be a partition of $S$ such that $M_j = \{s \in S | \mathcal{N}(s) \subseteq C_j, \mathcal{N}(s) \bigcap C_k = \emptyset \; \forall k < j\}$.

**Data**: Observations and parameters $(\mathbf{x}, \boldsymbol{\theta})$

**Output**: Likelihood and gradient $\left(\partial_{\mathbf{x}}[F(\mathbf{x}; \boldsymbol{\theta})], \nabla_{\boldsymbol{\theta}}\partial_{\mathbf{x}}[F(\mathbf{x}; \boldsymbol{\theta})]\right)$

**foreach** *Node* $j \in \mathcal{C}$ **do**
    $U_j \leftarrow \emptyset;\ \psi_j \leftarrow \prod_{s \in M_j} \phi_s$;

**1**    **foreach** *Subset* $A \subseteq C_j$ **do**
        $m_j(A, \emptyset) \leftarrow \partial_{\mathbf{x}_A}[\psi_j]$;
        $\mathbf{g}_j(A, \emptyset) \leftarrow \nabla_{\boldsymbol{\theta}}\partial_{\mathbf{x}_A}[\psi_j]$;
    **end**

**2**    **foreach** *Neighbor* $k \in \mathcal{E}_j \bigcap \tau_k^j$ **do**
        $S_{j,k} \leftarrow C_j \bigcap C_k$;
        **foreach** *Subset* $A \subseteq C_j$ **do**
            $m_j(A, U_j \bigcup k) \leftarrow \sum_{B \subseteq A \bigcap S_{j,k}} m_{k \to j}(B) m_j(A \setminus B, U_j)$;
            $\mathbf{g}_j(A, U_j \bigcup k) \leftarrow \sum_{B \subseteq A \bigcap S_{j,k}} m_{k \to j}(B)\mathbf{g}_j(A \setminus B, U_j) + \mathbf{g}_{k \to j}(B)m_j(A \setminus B, U_j)$;
        **end**
        $U_j \leftarrow U_j \bigcup k$;
    **end**
    **if** $j \neq r$ **then**
        $k \leftarrow \{l | \mathcal{E}_j \bigcap \tau_j^l \neq \emptyset\};\ S_{j,k} \leftarrow C_j \bigcap C_k$;

**3**        **foreach** *Subset* $A \subseteq S_{j,k}$ **do**
            $m_{j \to k}(A) \leftarrow m_j\left(A \bigcup C_j \setminus S_{j,k}, \mathcal{E}_j \setminus k\right)$;
            $\mathbf{g}_{j \to k}(A) \leftarrow \mathbf{g}_j\left(A \bigcup C_j \setminus S_{j,k}, \mathcal{E}_j \setminus k\right)$;
        **end**
    **else**
        **return** $\left(m_r(C_r, \mathcal{E}_r), \mathbf{g}_r(C_r, \mathcal{E}_r)\right)$
    **end**
**end**

---

Note that JDiff reduces to the algorithms of [6, 8] for non-loopy CDNs and its complexity then becomes linear in the number of variables. For other types of graphs, the complexity grows exponentially with the tree-width.

## 4 Experiments

The experiments are divided into two parts. The first part evaluates the computational efficiency of the JDiff algorithm for various graph topologies. The second set of experiments uses rainfall and H1N1 epidemiology data to demonstrate the practical value of loopy CDNs, which JDiff for the first time makes practical to learn from data.

### 4.1 Symbolic differentiation

As a first test, we compared the runtime of JDiff to that of commonly-used symbolic differentiation tools such as Mathematica [16] and D* [4]. The task here was to symbolically compute $\partial_{\mathbf{x}}[F(\mathbf{x})]$ for a variety of CDNs. All three algorithms were run on a machine with a 2.66 GHz CPU and 16 GB of RAM. The JDiff algorithm was implemented in MATLAB. A junction tree was constructed by greedily eliminating the variables with the minimal fill-in algorithm and then constructing elimination subsets for nodes in the junction tree [10] using the MATLAB implementation of [14]. For square grid-structured CDNs with CDN functions defined over pairs of adjacent variables, Mathematica and D* ran out of memory for grids larger than $3 \times 3$. For the $3 \times 3$ grid, JDiff took less than 1 second to compute the symbolic derivative, whereas Mathematica and D* took 6.2 s. and 9.2

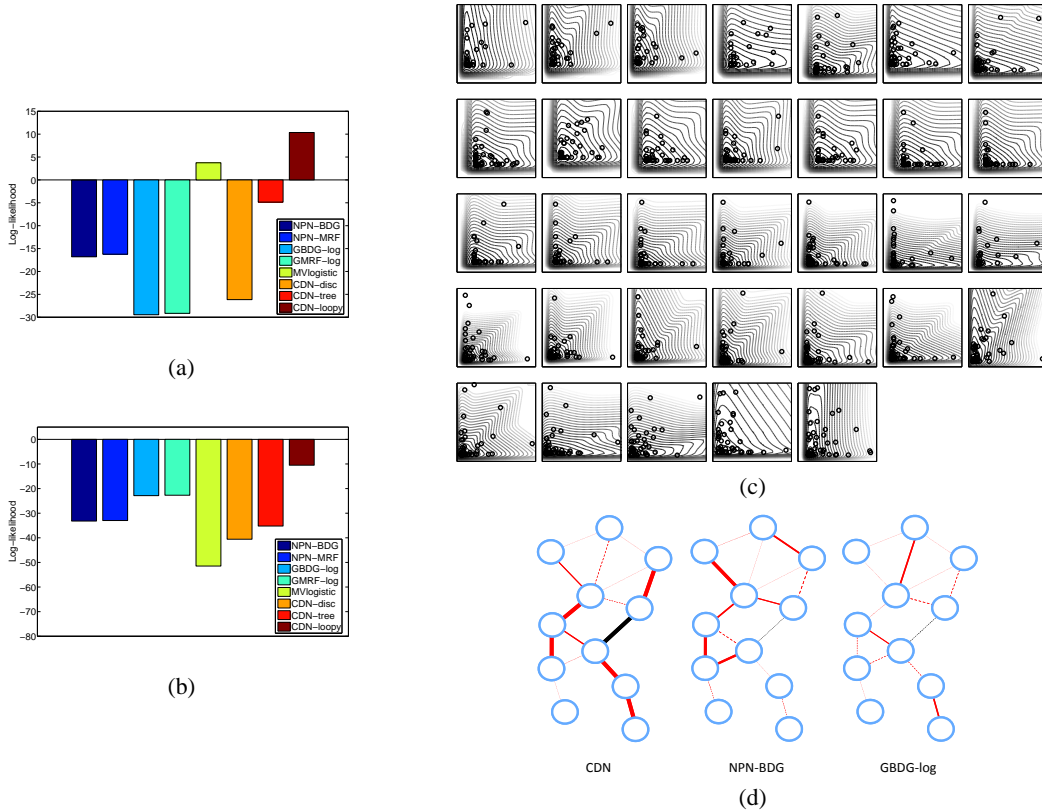

Figure 2: Both a), b) report average test log-likelihoods achieved for the CDNs, the nonparanormal bidirected and Markov models (NPN-BDG,NPN-MRF), Gaussian bidirected and Markov models for log-transformed data (GBDG-log,GMRF-log) and the multivariate logistic distribution (MVlogistic) on leave-one-out cross-validation of the a) rainfall and b) H1N1 datasets; c) Contour plots of log-bivariate densities under the CDN model of Figure 1(c) for rainfall with observed measurements shown. Each panel shows the marginal PDF $P(x_\alpha, x_\beta) = \partial_{x_{\alpha,\beta}}[F(x_\alpha, x_\beta)]$ under the CDN model for each CDN function $s$ and its neighbors $\alpha, \beta$. Each marginal PDF can be computed analytically by taking limits followed by differentiation; d) Graphs for the H1N1 datasets with edges weighted according to mutual information under the CDN, nonparanormal and Gaussian BDGs for log-transformed data. Dashed edges correspond to information of less than 1 bit.

s. each. We also found that JDiff could tractably (i.e.: in less than 20 min. of CPU time) compute derivatives for graphs as large as $9 \times 9$. We also compared the time to compute mixed derivatives in loops of length $n = 10, 11, \cdots, 20$. The time required by JDiff varied from $0.81$ s. to $2.83$ s. to compute the total mixed derivative, whereas the time required by Mathematica varied from $1.2$ s. to $580$ s. and for D*, $6.7$ s. to $12.7$ s.

### 4.2 Learning models for rainfall and H1N1 data

The JDiff algorithm allows us to compute mixed derivatives of a joint CDF for applications in which we may need to learn multivariate heavy-tailed distributions defined on loopy graphs. The graphical structures in our examples are based on geographical location of variables that impose dependence constraints based on spatial proximity. To model pairs of heavy-tailed variables, we used the bivariate logistic distribution with Gumbel margins [2], given by

$$\phi_s(x, y) = \exp\left(-\left(e^{-\frac{x-\mu_{x,s}}{\sigma_{x,s}\theta_s}} + e^{-\frac{y-\mu_{y,s}}{\sigma_{y,s}\theta_s}}\right)^{\theta_s}\right), \ \sigma_{x,s} > 0, \sigma_{y,s} > 0, \ 0 < \theta_s < 1. \quad (10)$$

Models constructed by computing products of functions of the above type have the properties of both being heavy-tailed multivariate distributions and satisfying marginal independence constraints between variables that share no function nodes [8]. Here we examined the data studied in [8], which consisted of spatial measurements for rainfall and for H1N1 mortality. The rainfall dataset consists of 61 daily measurements of rainfall at 22 sites in China and the H1N1 dataset consists of 29 weekly mortality rates in 11 cities in the Northeastern US during the 2008-2009 epidemic. Starting from the non-loopy CDNs used in [8] (Figures 1(c) and 1(d), shown in blue), we added function nodes and edges to construct loopy CDNs (shown in red in Figures 1(c) and 1(d)) to construct CDNs capable

of expressing many more marginal dependencies at the cost of creating numerous loops in the graph. All CDN models (non-loopy and loopy) were learned from data using stochastic gradients to update model parameters using settings described in the Supplemental Information.

The loopy CDN model was compared via leave-one-out cross-validation to non-loopy CDNs of [8] and disconnected CDNs corresponding to independence models. To compare with other multivariate approaches for modelling heavy-tailed data, we also tested the following:

- Gaussian bi-directed (BDG) and Markov (MRF) models with the same topology as the loopy CDNs for log-transformed data with $\tilde{x} = \log(x + \epsilon_i)$ for $\epsilon_i = 10^{-i}, i = 1, 2, 3, 4, 5$, where we show the results for $i$ that yielded the best test likelihood. Models were fitted using the algorithms of [3] and [15]. For the Gaussian BDGs, the covariance matrices $\Sigma$ were constrained so that $(\Sigma)_{\alpha,\beta} = 0$ only if there is no edge connecting variable nodes $\alpha, \beta$. For the Gaussian MRF, the constraints were $(\Sigma)^{-1}_{\alpha,\beta} = 0)$.

- Structured nonparanormal distributions [11], which use a Gaussian copula model, where the structure was specified by the same BDG and MRF graphs and estimation of the covariance was performed using the algorithms for Gaussian MRFs and BDGs on nonlinearly transformed data. The nonlinear transformation is given by $f_\alpha(x_\alpha) = \tilde{\mu}_\alpha + \tilde{\sigma}_\alpha \Phi^{-1}(\tilde{F}_\alpha(x_\alpha))$ where $\Phi$ is the normal CDF, $\tilde{F}_\alpha$ is the Winsorized estimator [11] of the CDF for random variable $X_\alpha$ and parameters $\tilde{\mu}_\alpha, \tilde{\sigma}_\alpha$ are the empirical mean and standard deviation for $X_\alpha$. Although the nonparanormal allows for structure learning as part of model fitting, for the sake of comparison the structure of the model was set to be same as those of the BDG and MRF models.

- The multivariate logistic CDF [13] that is heavy-tailed but does not model local dependencies.

Here we designed the BDG and MRF models to have the same graphical structure as the loopy CDN model such that all three model classes represent the same set of local dependencies even though the set of global dependencies is different for a BDG, MRF and CDN of the same connectivity. Additional details about these comparisons are provided in the Supplemental Information. The resulting average test log-likelihoods on leave-one-out cross-validation achieved by the above models are shown in Figures 2(a) and 2(b). Here, capturing the additional local dependencies and heavy-tailedness using loopy CDNs leads to significantly better fits ($p < 10^{-8}$, two-sided sign test).

To further explore the loopy CDN model, we can visualize the set of log-bivariate densities obtained from the loopy CDN model for the rainfall data in tandem with observed data (Figure 2(c)). The marginal bivariate density for each pair of neighboring variables is obtained by taking limits of the learned multivariate CDF and differentiating the resulting bivariate CDF. We can also examine the resulting models by comparing the mutual information (MI) between pairs of neighboring variables in the graphical models for the H1N1 dataset. This is shown in Figure 2(d) in the form of undirected weighted graphs where edges are weighted proportional to the MI between the two variable nodes connected by that edge. For the CDN, MI was computed by drawing 50,000 samples from the resulting density model via the Metropolis algorithm; for Gaussian models, the MI was obtained analytically. As can be seen, the loopy CDN model differs significantly from the nonparanormal and Gaussian BDGs for log-transformed data in the MI between pairs of variables (Figure 2(d)). Not only are the MI values under the loopy CDN model significantly higher as compared to those under the Gaussian models, but also high MI is assigned to the edge corresponding to the Newark,NJ/Philadelphia,PA air corridor, which is a likely source of H1N1 transmission between cities [1] (edge shown in black in Figure 2(d)). In contrast, this edge is largely missed by the nonparanormal and log-transformed Gaussian BDGs.

## 5 Discussion

The above results for the rainfall and H1N1 datasets, combined with the lower runtime of JDiff compared to standard symbolic differentiation algorithms, highlight A) the usefulness of JDiff as an algorithm for exact inference and learning for loopy CDNs and B) the usefulness of loopy CDNs in which multiple local functions can be used to model local dependencies between variables in the model. Future work could include learning the structure of compact probability models in the sense of graphs with bounded treewidth, with practical applications to other problem domains (e.g.: finance, seismology) in which data are heavy-tailed and high-dimensional and comparisons to existing techniques for doing this [11]. Another line of research would be to further study the connection between CDNs and other copula-based models (e.g.: [9]). Finally, given the demonstrated value of adding dependency constraints to CDNs, further development of faster approximate algorithms for loopy CDNs will also be of practical value.

# References

[1] Colizza, V., Barrat, A., Barthelemy, M. and Vespignani, A. (2006) Prediction and predictability of global epidemics: the role of the airline transportation network. *Proceedings of the National Academy of Sciences USA (PNAS)* **103**, 2015-2020.

[2] de Haan, L. and Ferreira, A. (2006) *Extreme value theory.* Springer.

[3] Drton, M. and Richardson, T.S. (2004) Iterative conditional fitting for Gaussian ancestral graph models. *Proceedings of the Twentieth Conference on Uncertainty in Artificial Intelligence (UAI)*, 130-137.

[4] Guenter, B. (2007) Efficient symbolic differentiation for graphics applications. *ACM Transactions on Graphics* **26(3)**.

[5] Hardy, M. (2006) Combinatorics of partial derivatives. *Electronic Journal of Combinatorics* **13**.

[6] Huang, J.C. and Frey, B.J. (2008) Cumulative distribution networks and the derivative-sum-product algorithm. *Proceedings of the Twenty-Fourth Conference on Uncertainty in Artificial Intelligence (UAI)*, 290-297.

[7] Huang, J.C. (2009) *Cumulative distribution networks: Inference, estimation and applications of graphical models for cumulative distribution functions.* University of Toronto Ph.D. thesis. http://hdl.handle.net/1807/19194

[8] Huang, J.C. and Jojic, N. (2010) Maximum-likelihood learning of cumulative distribution functions on graphs. *Journal of Machine Learning Research W&CP Series* **9**, 342-349.

[9] Kirschner, S. (2007) Learning with tree-averaged densities and distributions. *Advances in Neural Information Systems Processing (NIPS)* **20**, 761-768.

[10] Koller, D. and Friedman, N. (2009). *Probabilistic Graphical Models: Principles and Techniques,* MIT Press.

[11] Liu, H., Lafferty, J. and Wasserman, L. (2009) The nonparanormal: Semiparametric estimation of high dimensional undirected graphs. *Journal of Machine Learning Research (JMLR)* **10**, 2295-2328.

[12] Lauritzen, S.L. and Spiegelhalter, D.J. (1988) Local computations with probabilities on graphical structures and their application to expert systems. *Journal of the Royal Statistical Society Series B (Methodological)* **50(2)**, 157224.

[13] Malik, H.J. and Abraham, B. (1978) Multivariate logistic distributions. *Annals of Statistics* **1(3)**, 588-590.

[14] Murphy, K.P. (2001) The Bayes Net Toolbox for MATLAB. *Computing science and statistics*.

[15] Speed, T.S. and Kiiveri, H.T. (1986) Gaussian Markov distributions over finite graphs. *Annals of Statistics* **14(1)**, 138-150.

[16] Wolfram Research, Inc. (2008) Mathematica, Version 7.0. Champaign, IL.

